# A Bilinear Model for Sparse Coding

**David B. Grimes and Rajesh P. N. Rao**
Department of Computer Science and Engineering
University of Washington
Seattle, WA 98195-2350, U.S.A.
{grimes,rao}@cs.washington.edu

## Abstract

Recent algorithms for sparse coding and independent component analysis (ICA) have demonstrated how localized features can be learned from natural images. However, these approaches do not take image transformations into account. As a result, they produce image codes that are redundant because the same feature is learned at multiple locations. We describe an algorithm for sparse coding based on a bilinear generative model of images. By explicitly modeling the interaction between image features and their transformations, the bilinear approach helps reduce redundancy in the image code and provides a basis for transformation-invariant vision. We present results demonstrating bilinear sparse coding of natural images. We also explore an extension of the model that can capture spatial relationships between the independent features of an object, thereby providing a new framework for parts-based object recognition.

## 1 Introduction

Algorithms for redundancy reduction and efficient coding have been the subject of considerable attention in recent years [6, 3, 4, 7, 9, 5, 11]. Although the basic ideas can be traced to the early work of Attneave [1] and Barlow [2], recent techniques such as independent component analysis (ICA) and sparse coding have helped formalize these ideas and have demonstrated the feasibility of efficient coding through redundancy reduction. These techniques produce an efficient code by attempting to minimize the dependencies between elements of the code by using appropriate constraints.

One of the most successful applications of ICA and sparse coding has been in the area of image coding. Olshausen and Field showed that sparse coding of natural images produces localized, oriented basis filters that resemble the receptive fields of simple cells in primary visual cortex [6, 7]. Bell and Sejnowski obtained similar results using their algorithm for ICA [3]. However, these approaches do not take image transformations into account. As a result, the same oriented feature is often learned at different locations, yielding a redundant code. Moreover, the presence of the same feature at multiple locations prevents more complex features from being learned and leads to a combinatorial explosion when one attempts to scale the approach to large image patches or hierarchical networks.

In this paper, we propose an approach to sparse coding that explicitly models the interac-

tion between image features and their transformations. A bilinear generative model is used to learn both the independent features in an image as well as their transformations. Our approach extends Tenenbaum and Freeman's work on bilinear models for learning content and style [12] by casting the problem within probabilistic sparse coding framework. Thus, whereas prior work on bilinear models used global decomposition methods such as SVD, the approach presented here emphasizes the extraction of local features by removing higher-order redundancies through sparseness constraints. We show that for natural images, this approach produces localized, oriented filters that can be translated by different amounts to account for image features at arbitrary locations. Our results demonstrate how an image can be factored into a set of basic local features and their transformations, providing a basis for transformation-invariant vision. We conclude by discussing how the approach can be extended to allow parts-based object recognition, wherein an object is modeled as a collection of local features (or "parts") and their relative transformations.

## 2  Bilinear Generative Models

We begin by considering the standard linear generative model used in algorithms for ICA and sparse coding [3, 7, 9]:

$$\mathbf{z} = \sum_{i=1}^{m} \mathbf{w}_i x_i \tag{1}$$

where $\mathbf{z}$ is a $k$-dimensional input vector (e.g. an image), $\mathbf{w}_i$ is a $k$-dimensional basis vector and $x_i$ is its scalar coefficient. Given the linear generative model above, the goal of ICA is to learn the basis vectors $\mathbf{w}_i$ such that the $x_i$ are as independent as possible, while the goal in sparse coding is to make the distribution of $x_i$ highly kurtotic given Equation 1.

The linear generative model in Equation 1 can be extended to the bilinear case by using two independent sets of coefficients $x_i$ and $y_i$ (or equivalently, two vectors $\mathbf{x}$ and $\mathbf{y}$) [12]:

$$\mathbf{z} = f(\mathbf{x}, \mathbf{y}) = \sum_{i=1}^{m} \sum_{j=1}^{n} \mathbf{w}_{ij} x_i y_j \tag{2}$$

The coefficients $x_i$ and $y_j$ jointly modulate a set of basis vectors $\mathbf{w}_{ij}$ to produce an input vector $\mathbf{z}$. For the present study, the coefficient $x_i$ can be regarded as encoding the presence of object feature $i$ in the image while the $y_j$ values determine the transformation present in the image. In the terminology of Tenenbaum and Freeman [12], $\mathbf{x}$ describes the "content" of the image while $\mathbf{y}$ encodes its "style."

Equation 2 can also be expressed as a linear equation in $\mathbf{x}$ for a fixed $\mathbf{y}$:

$$\mathbf{z} = f(\mathbf{x})_{\mathbf{y}} = \sum_{i=1}^{m} \left( \sum_{j=1}^{n} \mathbf{w}_{ij} y_j \right) x_i = \sum_{i=1}^{m} \mathbf{w}_i^y x_i \tag{3}$$

Likewise, for a fixed $\mathbf{x}$, one obtains a linear equation in $\mathbf{x}$. Indeed this is the definition of bilinear: given one fixed factor, the model is linear with respect to the other factor. The power of bilinear models stems from the rich non-linear interactions that can be represented by varying both $\mathbf{x}$ and $\mathbf{y}$ simultaneously.

## 3  Learning Sparse Bilinear Models

### 3.1  Learning Bilinear Models

Our goal is to learn from image data an appropriate set of basis vectors $\mathbf{w}_{ij}$ that effectively describe the interactions between the feature vector $\mathbf{x}$ and the transformation vector $\mathbf{y}$.

A commonly used approach in unsupervised learning is to minimize the sum of squared pixel-wise errors over all images:

$$E_1(\mathbf{w}_{ij}, \mathbf{x}, \mathbf{y}) \quad = \quad ||\mathbf{z} - \sum_{i=1}^{m}\sum_{j=1}^{n}\mathbf{w}_{ij}x_iy_j||^2 \tag{4}$$

$$= \quad (\mathbf{z} - \sum_{i=1}^{m}\sum_{j=1}^{n}\mathbf{w}_{ij}x_iy_j)^T(\mathbf{z} - \sum_{i=1}^{m}\sum_{j=1}^{n}\mathbf{w}_{ij}x_iy_j) \tag{5}$$

where $|| \cdot ||$ denotes the $L_2$ norm of a vector. A standard approach to minimizing such a function is to use gradient descent and alternate between minimization with respect to $\{\mathbf{x}, \mathbf{y}\}$ and minimization with respect to $\mathbf{w}_{ij}$. Unfortunately, the optimization problem as stated is underconstrained. The function $E_1$ has many local minima and results from our simulations indicate that convergence is difficult in many cases. There are many different ways to represent an image, making it difficult for the method to converge to a basis set that can generalize effectively.

A related approach is presented by Tenenbaum and Freeman [12]. Rather than using gradient descent, their method estimates the parameters directly by computing the singular value decomposition (SVD) of a matrix $A$ containing input data corresponding to each content class $\mathbf{x}$ in every style $\mathbf{y}$. Their approach can be regarded as an extension of methods based on principal component analysis (PCA) applied to the bilinear case. The SVD approach avoids the difficulties of convergence that plague the gradient descent method and is much faster in practice. Unfortunately, the learned features tend to be global and non-localized similar to those obtained from PCA-based methods based on second-order statistics. As a result, the method is unsuitable for the problem of learning local features of objects and their transformations.

The underconstrained nature of the problem can be remedied by imposing constraints on $\mathbf{x}$ and $\mathbf{y}$. In particular, we could cast the problem within a probabilistic framework and impose specific prior distributions on $\mathbf{x}$ and $\mathbf{y}$ with higher probabilities for values that achieve certain desirable properties. We focus here on the class of sparse prior distributions for several reasons: (a) by forcing most of the coefficients to be zero for any given input, sparse priors minimize redundancy and encourage statistical independence between the various $x_i$ and between the various $y_j$ [7], (b) there is growing evidence for sparse representations in the brain – the distribution of neural responses in visual cortical areas is highly kurtotic i.e. the cell exhibits little activity for most inputs but responds vigorously for a few inputs, causing a distribution with a high peak near zero and long tails, (c) previous approaches based on sparseness constraints have obtained encouraging results [7], and (d) enforcing sparseness on the $x_i$ encourages the parts and local features shared across objects to be learned while imposing sparseness on the $y_j$ allows object transformations to be explained in terms of a small set of basic transformations.

## 3.2 Bilinear Sparse Coding

We assume the following priors for $x_i$ and $y_j$:

$$P(x_i) \quad = \quad \frac{1}{Z_\alpha}e^{-\alpha S(x_i)} \tag{6}$$

$$P(y_j) \quad = \quad \frac{1}{Z_\beta}e^{-\beta S(y_j)} \tag{7}$$

where $Z_\alpha$ and $Z_\beta$ are normalization constants, $\alpha$ and $\beta$ are parameters that control the degree of sparseness, and $S$ is a "sparseness function." For this study, we used $S(a) = \log(1 + a^2)$.

Within a probabilistic framework, the squared error function $E_1$ summed over all images can be interpreted as representing the negative log likelihood of the data given the parameters: $-\log P(\mathbf{z}|\mathbf{w}_{ij}, \mathbf{x}, \mathbf{y})$ (see, for example, [7]). The priors $P(x_i)$ and $P(y_j)$ can be used to marginalize this likelihood to obtain the new likelihood function: $L(\mathbf{w}_{ij}) = P(\mathbf{z}|\mathbf{w}_{ij})$. The goal then is to find the $\mathbf{w}_{ij}$ that maximize $L$, or equivalently, minimize the negative log of $L$. Under certain reasonable assumptions (discussed in [7]), this is equivalent to minimizing the following optimization function over all input images:

$$E(\mathbf{w}_{ij}, \mathbf{x}, \mathbf{y}) \quad = \quad ||\mathbf{z} - \sum_{i=1}^{m}\sum_{j=1}^{n}\mathbf{w}_{ij}x_iy_j||^2 + \alpha\sum_{i=1}^{m}S(x_i) + \beta\sum_{j=1}^{n}S(y_j) \qquad (8)$$

Gradient descent can be used to derive update rules for the components $x_a$ and $y_b$ of the feature vector $\mathbf{x}$ and transformation vector $\mathbf{y}$ respectively for any image $\mathbf{z}$, assuming a fixed basis $\mathbf{w}_{ij}$:

$$\frac{dx_a}{dt} = -\frac{1}{2}\frac{\partial E}{\partial x_a} \quad = \quad \sum_{q=1}^{n}\mathbf{w}_{aq}^T(\mathbf{z} - \sum_{i=1}^{m}\sum_{j=1}^{n}\mathbf{w}_{ij}x_iy_j)y_q + \frac{\alpha}{2}S'(x_a) \qquad (9)$$

$$\frac{dy_b}{dt} = -\frac{1}{2}\frac{\partial E}{\partial y_b} \quad = \quad \sum_{q=1}^{m}\mathbf{w}_{qb}^T(\mathbf{z} - \sum_{i=1}^{m}\sum_{j=1}^{n}\mathbf{w}_{ij}x_iy_j)x_q + \frac{\beta}{2}S'(y_b) \qquad (10)$$

Given a training set of inputs $\mathbf{z}_l$, the values for $\mathbf{x}$ and $\mathbf{y}$ for each image after convergence can be used to update the basis set $\mathbf{w}_{ij}$ in batch mode according to:

$$\frac{d\mathbf{w}_{ab}}{dt} = -\frac{1}{2}\frac{\partial E}{\partial \mathbf{w}_{ab}} \quad = \quad \sum_{l=1}^{M}(\mathbf{z}_l - \sum_{i=1}^{m}\sum_{j=1}^{n}\mathbf{w}_{ij}x_iy_j)x_ay_b \qquad (11)$$

As suggested by Olshausen and Field [7], in order to keep the basis vectors from growing without bound, we adapted the $L_2$ norm of each basis vector in such a way that the variances of the $x_i$ and $y_j$ were maintained at a fixed desired level.

## 4 Results

### 4.1 Training Paradigm

We tested the algorithms for bilinear sparse coding on natural image data. The natural images we used are distributed by Olshausen and Field [7], along with the code for their algorithm. The training set of images consisted of $10 \times 10$ patches randomly extracted from ten $512 \times 512$ source images. The images are pre-whitened to equalize large variances in frequency, and thus speed convergence. We choose to use a complete basis where $m = 100$ and we let $n$ be at least as large as the number of transformations (including the no-transformation case). The sparseness parameters $\alpha$ and $\beta$ were set to 2.2 and 1.5. In order to assist convergence all learning occurs in batch mode, where the batch consisted of $M = 100$ image patches. The step size $\eta$ for gradient descent using Equation 11 was set to 0.05. The transformations were chosen to be 2D translations in the range $[-4 : 4]$ pixels in both the axes. The style/content separation was enforced by learning a single $\mathbf{x}$ vector to describe an image patch regardless its translation, and likewise a single $\mathbf{y}$ vector to describe a particular style given any image patch content.

### 4.2 Bilinear Sparse Coding of Natural Images

Figure 1 shows the results of training on natural image data. A comparison between the learned features for the linear generative model (Equation 1) and the bilinear model is

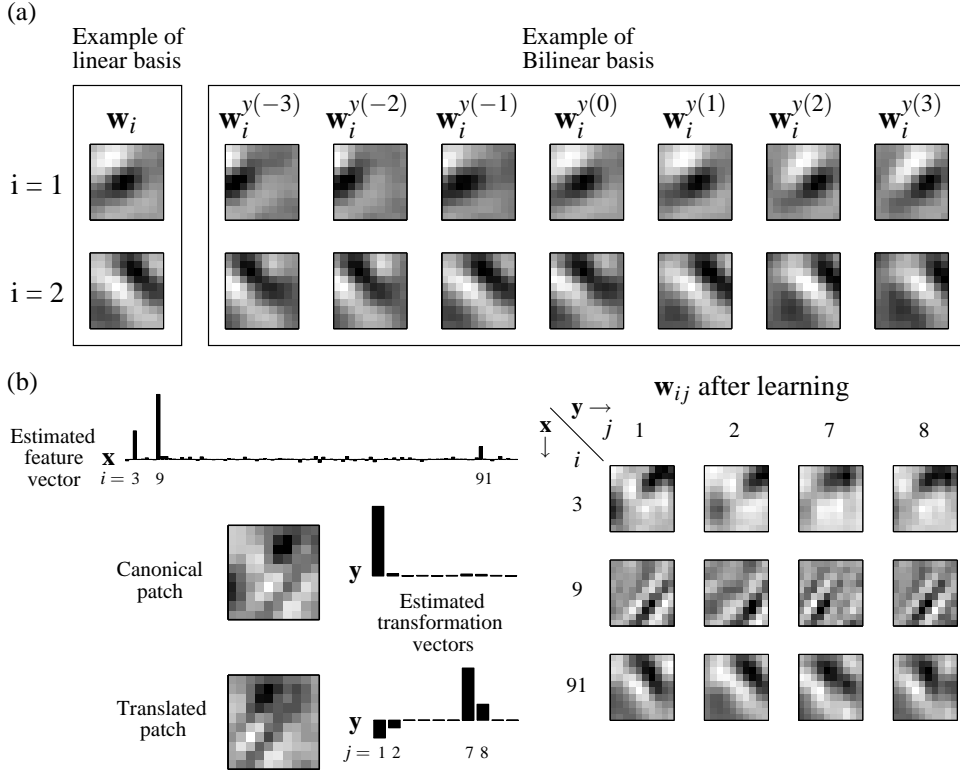

Figure 1: **Representing natural images and their transformations with a sparse bilinear model.** (a) A comparison of learned features between a standard linear model and a bilinear model, both trained with the same sparseness priors. The two rows for the bilinear case depict the translated object features $\mathbf{w}_i^y$ (see Equation 3) for translations of $-3, \ldots, 3$ pixels. (b) The representation of an example natural image patch, and of the same patch translated to the left. Note that the bar plot representing the $\mathbf{x}$ vector is indeed sparse, having only three significant coefficients. The code for the style vectors for both the canonical patch, and the translated one is likewise sparse. The $\mathbf{w}_{ij}$ basis images are shown for those dimensions which have non-zero coefficients for $x_i$ or $y_j$.

provided in Figure 1 (a). Although both show simple, localized, and oriented features, the bilinear method is able to model the same features under different transformations. In this case, the range $[-3, 3]$ horizontal translations were used in the training of the bilinear model. Figure 1 (b) provides an example of how the bilinear sparse coding model encodes a natural image patch and the same patch after it has been translated. Note that both the $\mathbf{x}$ and $\mathbf{y}$ vectors are sparse.

Figure 2 shows how the model can account for a given localized feature at different locations by varying the $\mathbf{y}$ vector. As shown in the last column of the figure, the translated local feature is generated by linearly combining a sparse set of basis vectors $\mathbf{w}_{ij}$.

## 4.3 Towards Parts-Based Object Recognition

The bilinear generative model in Equation 2 uses the same set of transformation values $y_j$ for all the features $i = 1, \ldots, m$. Such a model is appropriate for global transformations

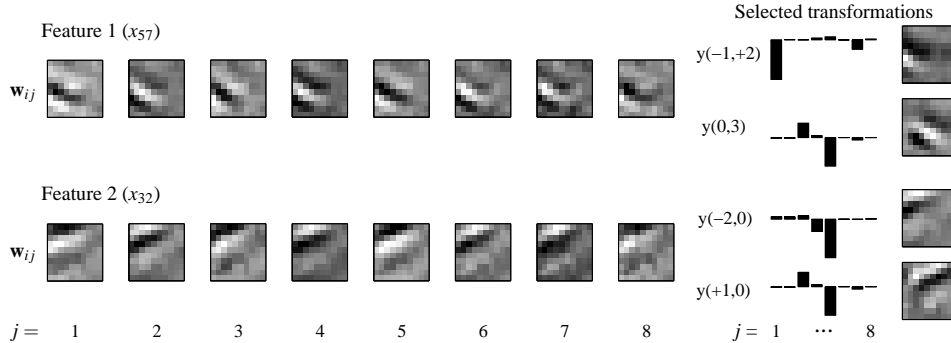

Figure 2: **Translating a learned feature to multiple locations.** The two rows of eight images represent the individual basis vectors $\mathbf{w}_{ij}$ for two values of $i$. The $y_j$ values for two selected transformations for each $i$ are shown as bar plots. $y(a, b)$ denotes a translation of $(a, b)$ pixels in the Cartesian plane. The last column shows the resulting basis vectors after translation.

that apply to an entire image region such as a shift of $p$ pixels for an image patch or a global illumination change.

Consider the problem of representing an object in terms of its constituent parts. In this case, we would like to be able to transform each part independently of other parts in order to account for the location, orientation, and size of each part in the object image. The standard bilinear model can be extended to address this need as follows:

$$\mathbf{z} = \sum_{i=1}^{m} \left( \sum_{j=1}^{n} \mathbf{w}_{ij} y_j^i \right) x_i \qquad (12)$$

Note that each object feature $i$ now has its own set of transformation values $y_j^i$. The double summation is thus no longer symmetric. Also note that the standard model (Equation 2) is a special case of Equation 12 where $y_j^i = y_j$ for all $i$.

We have conducted preliminary experiments to test the feasibility of Equation 12 using a set of object features learned for the standard bilinear model. Fig. 3 shows the results. These results suggest that allowing independent transformations for the different features provides a rich substrate for modeling images and objects in terms of a set of local features (or parts) and their individual transformations.

## 5 Summary and Conclusion

A fundamental problem in vision is to simultaneously recognize objects and their transformations [8, 10]. Bilinear generative models provide a tractable way of addressing this problem by factoring an image into object features and transformations using a bilinear equation. Previous approaches used unconstrained bilinear models and produced global basis vectors for image representation [12]. In contrast, recent research on image coding has stressed the importance of localized, independent features derived from metrics that emphasize the higher-order statistics of inputs [6, 3, 7, 5]. This paper introduces a new probabilistic framework for learning bilinear generative models based on the idea of sparse coding.

Our results demonstrate that bilinear sparse coding of natural images produces localized oriented basis vectors that can simultaneously represent features in an image and their transformation. We showed how the learned generative model can be used to translate a

(a)

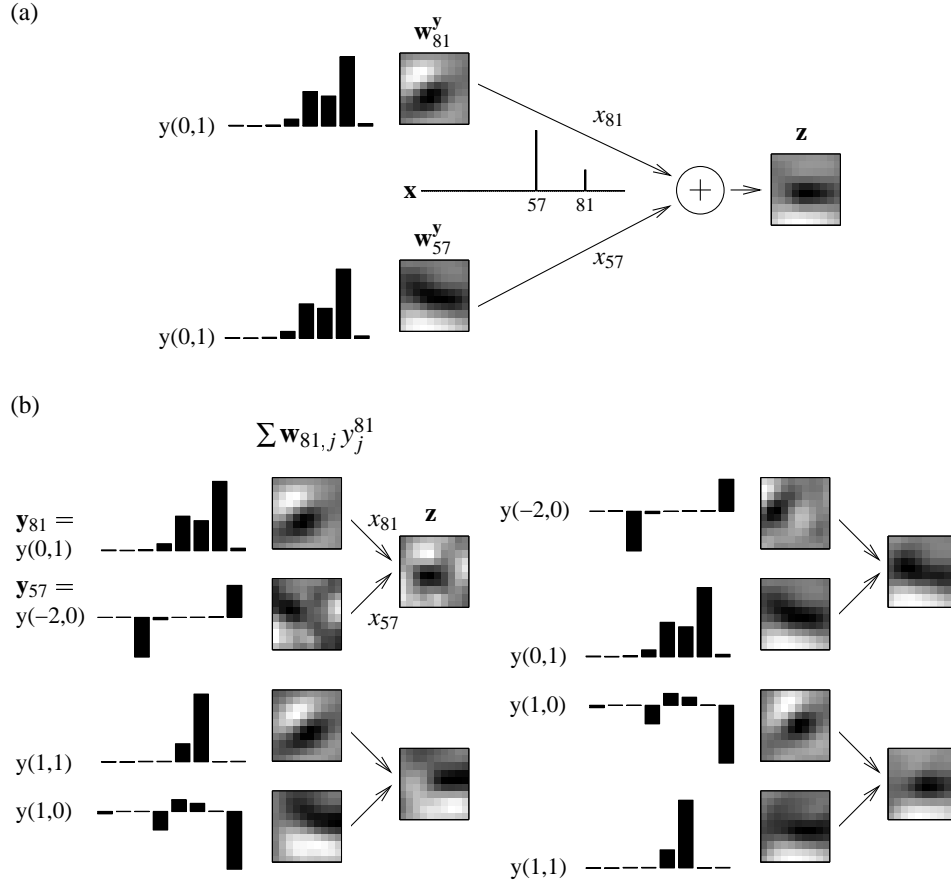

(b)

Figure 3: **Modeling independently transformed features.** (a) shows the standard bilinear method of generating a translated feature by combining basis vectors $\mathbf{w}_{ij}$ using the same set of $y_j$ values for two different features ($i = 57$ and $81$). (b) shows four examples of images generated by allowing different values of $y_j$ for the two different features. Note the significant differences between the resulting images, which cannot be obtained using the standard bilinear model.

basis vector to different locations, thereby reducing the need to learn the same basis vector at multiple locations as in traditional sparse coding methods. We also proposed an extension of the bilinear model that allows each feature to be transformed independently of other features. Our preliminary results suggest that such an approach could provide a flexible platform for adaptive parts-based object recognition, wherein objects are described by a set of independent, shared parts and their transformations. The importance of parts-based methods has long been recognized in object recognition in view of their ability to handle a combinatorially large number of objects by combining parts and their transformations. Few methods, if any, exist for learning representations of object parts and their transformations directly from images. Our ongoing efforts are therefore focused on deriving efficient algorithms for parts-based object recognition based on the combination of bilinear models and sparse coding.

**Acknowledgments**

This research is supported by NSF grant no. 133592 and a Sloan Research Fellowship to RPNR.

## References

[1] F. Attneave. Some informational aspects of visual perception. *Psychological Review*, 61(3):183–193, 1954.

[2] H. B. Barlow. Possible principles underlying the transformation of sensory messages. In W. A. Rosenblith, editor, *Sensory Communication*, pages 217–234. Cambridge, MA: MIT Press, 1961.

[3] A. J. Bell and T. J. Sejnowski. The 'independent components' of natural scenes are edge filters. *Vision Research*, 37(23):3327–3338, 1997.

[4] G. E. Hinton and Z. Ghahramani. Generative models for discovering sparse distributed representations. *Philosophical Transactions Royal Society B*, 352(1177–1190), 1997.

[5] M. S. Lewicki and T. J. Sejnowski. Learning overcomplete representations. *Neural Computation*, 12(2):337–365, 2000.

[6] B. A. Olshausen and D. J. Field. Emergence of simple-cell receptive field properties by learning a sparse code for natural images. *Nature*, 381:607–609, 1996.

[7] B. A. Olshausen and D. J. Field. Sparse coding with an overcomplete basis set: A strategy employed by V1? *Vision Research*, 37:33113325, 1997.

[8] R. P. N. Rao and D. H. Ballard. Development of localized oriented receptive fields by learning a translation-invariant code for natural images. *Network: Computation in Neural Systems*, 9(2):219–234, 1998.

[9] R. P. N. Rao and D. H. Ballard. Predictive coding in the visual cortex: A functional interpretation of some extra-classical receptive field effects. *Nature Neuroscience*, 2(1):79–87, 1999.

[10] R. P. N. Rao and D. L. Ruderman. Learning Lie groups for invariant visual perception. In *Advances in Neural Information Processing Systems 11*, pages 810–816. Cambridge, MA: MIT Press, 1999.

[11] O. Schwartz and E. P. Simoncelli. Natural signal statistics and sensory gain control. *Nature Neuroscience*, 4(8):819–825, August 2001.

[12] J. B. Tenenbaum and W. T. Freeman. Separating style and content with bilinear models. *Neural Computation*, 12(6):1247–1283, 2000.